# Reading Tea Leaves: How Humans Interpret Topic Models

**Jonathan Chang** [*]
Facebook
1601 S California Ave.
Palo Alto, CA 94304
jonchang@facebook.com

**Jordan Boyd-Graber** [*]
Institute for Advanced Computer Studies
University of Maryland
jbg@umiacs.umd.edu

**Sean Gerrish, Chong Wang, David M. Blei**
Department of Computer Science
Princeton University
{sgerrish,chongw,blei}@cs.princeton.edu

## Abstract

Probabilistic topic models are a popular tool for the unsupervised analysis of text, providing both a predictive model of future text and a latent topic representation of the corpus. Practitioners typically assume that the latent space is semantically meaningful. It is used to check models, summarize the corpus, and guide exploration of its contents. However, whether the latent space is interpretable is in need of quantitative evaluation. In this paper, we present new quantitative methods for measuring semantic meaning in inferred topics. We back these measures with large-scale user studies, showing that they capture aspects of the model that are undetected by previous measures of model quality based on held-out likelihood. Surprisingly, topic models which perform better on held-out likelihood may infer less semantically meaningful topics.

## 1   Introduction

Probabilistic topic models have become popular tools for the unsupervised analysis of large document collections [1]. These models posit a set of latent *topics*, multinomial distributions over words, and assume that each document can be described as a mixture of these topics. With algorithms for fast approximate posterior inference, we can use topic models to discover both the topics and an assignment of topics to documents from a collection of documents. (See Figure 1.)

These modeling assumptions are useful in the sense that, empirically, they lead to good models of documents. They also anecdotally lead to semantically meaningful decompositions of them: topics tend to place high probability on words that represent concepts, and documents are represented as expressions of those concepts. Perusing the inferred topics is effective for model verification and for ensuring that the model is capturing the practitioner's intuitions about the documents. Moreover, producing a human-interpretable decomposition of the texts can be a goal in itself, as when browsing or summarizing a large collection of documents.

In this spirit, much of the literature comparing different topic models presents examples of topics and examples of document-topic assignments to help understand a model's mechanics. Topics also can help users discover new content via corpus exploration [2]. The presentation of these topics serves, either explicitly or implicitly, as a qualitative evaluation of the latent space, but there is no explicit *quantitative* evaluation of them. Instead, researchers employ a variety of metrics of model fit, such as perplexity or held-out likelihood. Such measures are useful for evaluating the predictive model, but do not address the more explatory goals of topic modeling.

---

[*]Work done while at Princeton University.

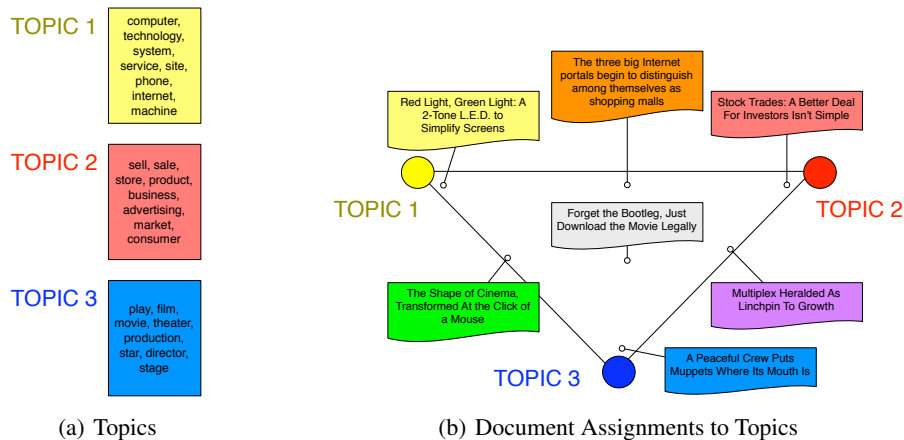

(a) Topics  (b) Document Assignments to Topics

Figure 1: The latent space of a topic model consists of topics, which are distributions over words, and a distribution over these topics for each document. On the left are three topics from a fifty topic LDA model trained on articles from the New York Times. On the right is a simplex depicting the distribution over topics associated with seven documents. The line from each document's title shows the document's position in the topic space.

In this paper, we present a method for measuring the interpretatability of a topic model. We devise two human evaluation tasks to explicitly evaluate both the quality of the topics inferred by the model and how well the model assigns topics to documents. The first, *word intrusion*, measures how semantically "cohesive" the topics inferred by a model are and tests whether topics correspond to natural groupings for humans. The second, *topic intrusion*, measures how well a topic model's decomposition of a document as a mixture of topics agrees with human associations of topics with a document. We report the results of a large-scale human study of these tasks, varying both modeling assumptions and number of topics. We show that these tasks capture aspects of topic models not measured by existing metrics and–surprisingly–models which achieve better predictive perplexity often have less interpretable latent spaces.

## 2   Topic models and their evaluations

Topic models posit that each document is expressed as a mixture of topics. These topic proportions are drawn once per document, and the topics are shared across the corpus. In this paper we will consider topic models that make different assumptions about the topic proportions. Probabilistic Latent Semantic Indexing (pLSI) [3] makes no assumptions about the document topic distribution, treating it as a distinct parameter for each document. Latent Dirichlet allocation (LDA) [4] and the correlated topic model (CTM) [5] treat each document's topic assignment as a multinomial random variable drawn from a symmetric Dirichlet and logistic normal prior, respectively.

While the models make different assumptions, inference algorithms for all of these topic models build the same type of latent space: a collection of topics for the corpus and a collection of topic proportions for each of its documents. While this common latent space has explored for over two decades, its interpretability remains unmeasured.

**Pay no attention to the latent space behind the model**

Although we focus on probabilistic topic models, the field began in earnest with latent semantic analysis (LSA) [6]. LSA, the basis of pLSI's probabilistic formulation, uses linear algebra to decompose a corpus into its constituent themes. Because LSA originated in the psychology community, early evaluations focused on replicating human performance or judgments using LSA: matching performance on standardized tests, comparing sense distinctions, and matching intuitions about synonymy (these results are reviewed in [7]). In information retrieval, where LSA is known as latent semantic indexing (LSI) [8], it is able to match queries to documents, match experts to areas of expertise, and even generalize across languages given a parallel corpus [9].

The reticence to look under the hood of these models has persisted even as models have moved from psychology into computer science with the development of pLSI and LDA. Models either use measures based on held-out likelihood [4, 5] or an external task that is independent of the topic space such as sentiment detection [10] or information retrieval [11]. This is true even for models engineered to have semantically coherent topics [12].

For models that use held-out likelihood, Wallach et al. [13] provide a summary of evaluation techniques. These metrics borrow tools from the language modeling community to measure how well the information learned from a corpus applies to unseen documents. These metrics generalize easily and allow for likelihood-based comparisons of different models or selection of model parameters such as the number of topics. However, this adaptability comes at a cost: these methods only measure the probability of observations; the internal representation of the models is ignored.

Griffiths et al. [14] is an important exception to the trend of using external tasks or held-out likelihood. They showed that the number of topics a word appears in correlates with how many distinct senses it has and reproduced many of the metrics used in the psychological community based on human performance. However, this is still not a deep analysis of the structure of the latent space, as it does not examine the structure of the topics themselves.

We emphasize that not measuring the internal representation of topic models is at odds with their presentation and development. Most topic modeling papers display qualitative assessments of the inferred topics or simply assert that topics are semantically meaningful, and practitioners use topics for model checking during the development process. Hall et al. [15], for example, used latent topics deemed historically relevant to explore themes in the scientific literature. Even in production environments, topics are presented as themes: Rexa (http://rexa.info), a scholarly publication search engine, displays the topics associated with documents. This implicit notion that topics have semantic meaning for users has even motivated work that attempts to automatically label topics [16]. Our goal is to measure the success of interpreting topic models across number of topics and modeling assumptions.

## 3 Using human judgments to examine the topics

Although there appears to be a longstanding assumption that the latent space discovered by topic models is meaningful and useful, evaluating such assumptions is difficult because discovering topics is an unsupervised process. There is no gold-standard list of topics to compare against for every corpus. Thus, evaluating the latent space of topic models requires us to gather exogenous data.

In this section we propose two tasks that create a formal setting where humans can evaluate the two components of the latent space of a topic model. The first component is the makeup of the topics. We develop a task to evaluate whether a topic has human-identifiable semantic coherence. This task is called *word intrusion*, as subjects must identify a spurious word inserted into a topic. The second task tests whether the association between a document and a topic makes sense. We call this task *topic intrusion*, as the subject must identify a topic that was not associated with the document by the model.

### 3.1 Word intrusion

To measure the coherence of these topics, we develop the *word intrusion* task; this task involves evaluating the latent space presented in Figure 1(a). In the word intrusion task, the subject is presented with six randomly ordered words. The task of the user is to find the word which is out of place or does not belong with the others, i.e., the *intruder*. Figure 2 shows how this task is presented to users.

When the set of words minus the intruder makes sense together, then the subject should easily identify the intruder. For example, most people readily identify `apple` as the intruding word in the set {`dog, cat, horse, apple, pig, cow`} because the remaining words, {`dog, cat, horse, pig, cow`} make sense together — they are all animals. For the set {`car, teacher, platypus, agile, blue, Zaire`}, which lacks such coherence, identifying the intruder is difficult. People will typically choose an intruder at random, implying a topic with poor coherence.

In order to construct a set to present to the subject, we first select at random a topic from the model. We then select the five most probable words from that topic. In addition to these words, an intruder

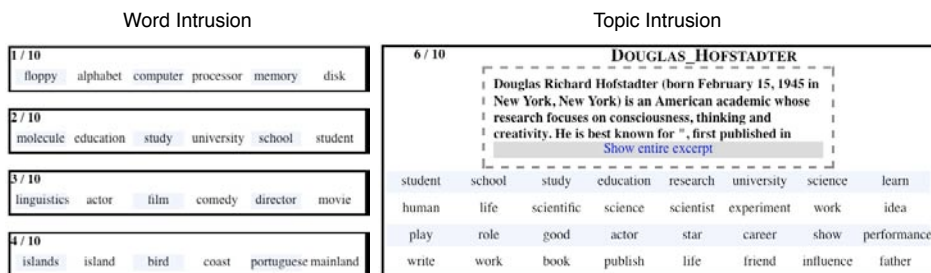

Figure 2: Screenshots of our two human tasks. In the word intrusion task (left), subjects are presented with a set of words and asked to select the word which does not belong with the others. In the *topic intrusion* task (right), users are given a document's title and the first few sentences of the document. The users must select which of the four groups of words does not belong.

word is selected at random from a pool of words with low probability in the current topic (to reduce the possibility that the intruder comes from the same semantic group) but high probability in some other topic (to ensure that the intruder is not rejected outright due solely to rarity). All six words are then shuffled and presented to the subject.

## 3.2 Topic intrusion

The *topic intrusion* task tests whether a topic model's decomposition of documents into a mixture of topics agrees with human judgments of the document's content. This allows for evaluation of the latent space depicted by Figure 1(b). In this task, subjects are shown the title and a snippet from a document. Along with the document they are presented with four topics (each topic is represented by the eight highest-probability words within that topic). Three of those topics are the highest probability topics assigned to that document. The remaining *intruder topic* is chosen randomly from the other low-probability topics in the model.

The subject is instructed to choose the topic which does not belong with the document. As before, if the topic assignment to documents were relevant and intuitive, we would expect that subjects would select the topic we randomly added as the topic that did not belong. The formulation of this task provides a natural way to analyze the quality of document-topic assignments found by the topic models. Each of the three models we fit explicitly assigns topic weights to each document; this task determines whether humans make the same association.

Due to time constraints, subjects do not see the entire document; they only see the title and first few sentences. While this is less information than is available to the algorithm, humans are good at extrapolating from limited data, and our corpora (encyclopedia and newspaper) are structured to provide an overview of the article in the first few sentences. The setup of this task is also meaningful in situations where one might be tempted to use topics for corpus exploration. If topics are used to find relevant documents, for example, users will likely be provided with similar views of the documents (e.g. title and abstract, as in Rexa).

For both the word intrusion and topic intrusion tasks, subjects were instructed to focus on the meanings of words, not their syntactic usage or orthography. We also presented subjects with the option of viewing the "correct" answer after they submitted their own response, to make the tasks more engaging. Here the "correct" answer was determined by the model which generated the data, presented as if it were the response of another user. At the same time, subjects were encouraged to base their responses on their own opinions, not to try to match other subjects' (the models') selections. In small experiments, we have found that this extra information did not bias subjects' responses.

## 4 Experimental results

To prepare data for human subjects to review, we fit three different topic models on two corpora. In this section, we describe how we prepared the corpora, fit the models, and created the tasks described in Section 3. We then present the results of these human trials and compare them to metrics traditionally used to evaluate topic models.

### 4.1 Models and corpora

In this work we study three topic models: probabilistic latent semantic indexing (pLSI) [3], latent Dirichlet allocation (LDA) [4], and the correlated topic model (CTM) [5], which are all mixed membership models [17]. The number of latent topics, $K$, is a free parameter in each of the models; here we explore this with $K = 50$, $100$ and $150$. The remaining parameters – $\beta_k$, the topic multinomial distribution for topic $k$; and $\theta_d$, the topic mixture proportions for document $d$ – are inferred from data. The three models differ in how these latent parameters are inferred.

**pLSI** In pLSI, the topic mixture proportions $\theta_d$ are a parameter for each document. Thus, pLSI is not a fully generative model, and the number of parameters grows linearly with the number of documents. We fit pLSI using the EM algorithm [18] but regularize pLSI's estimates of $\theta_d$ using pseudo-count smoothing, $\alpha = 1$.

**LDA** LDA is a fully generative model of documents where the mixture proportions $\theta_d$ are treated as a random variable drawn from a Dirichlet prior distribution. Because the direct computation of the posterior is intractable, we employ variational inference [4] and set the symmetric Dirichlet prior parameter, $\alpha$, to 1.

**CTM** In LDA, the components of $\theta_d$ are nearly independent (i.e., $\theta_d$ is statistically neutral). CTM allows for a richer covariance structure between topic proportions by using a logistic normal prior over the topic mixture proportions $\theta_d$. For each topic, $k$, a real $\gamma$ is drawn from a normal distribution and exponentiated. This set of $K$ non-negative numbers are then normalized to yield $\theta_d$. Here, we train the CTM using variational inference [5].

We train each model on two corpora. For each corpus, we apply a part of speech tagger [19] and remove all tokens tagged as proper nouns (this was for the benefit of the human subjects; success in early experiments required too much encyclopedic knowledge). Stop words [20] and terms occurring in fewer than five documents are also removed. The two corpora we use are 1.) a collection of 8447 articles from the *New York Times* from the years 1987 to 2007 with a vocabulary size of 8269 unique types and around one million tokens and 2.) a sample of 10000 articles from *Wikipedia* (http://www.wikipedia.org) with a vocabulary size of 15273 unique types and three million tokens.

### 4.2 Evaluation using conventional objective measures

There are several metrics commonly used to evaluate topic models in the literature [13]. Many of these metrics are *predictive* metrics; that is, they capture the model's ability to predict a *test set* of unseen documents after having learned its parameters from a *training set*. In this work, we set aside 20% of the documents in each corpus as a test set and train on the remaining 80% of documents. We then compute predictive rank and predictive log likelihood.

To ensure consistency of evaluation across different models, we follow Teh et al.'s [21] approximation of the predictive likelihood $p(\mathbf{w}_d|D_{\text{train}})$ using $p(\boldsymbol{w}_d|D_{\text{train}}) \approx p(\boldsymbol{w}_d|\hat{\theta}_d)$, where $\hat{\theta}_d$ is a point estimate of the posterior topic proportions for document $d$. For pLSI $\hat{\theta}_d$ is the MAP estimate; for LDA and CTM $\hat{\theta}_d$ is the mean of the variational posterior. With this information, we can ask what words the model believes will be in the document and compare it with the document's actual composition. Given document $\boldsymbol{w}_d$, we first estimate $\hat{\theta}_d$ and then for every word in the vocabulary, we compute $p(w|\hat{\theta}_d) = \sum_z p(w|z)p(z|\hat{\theta}_d)$. Then we compute the average rank for the terms that actually appeared in document $\boldsymbol{w}_d$ (we follow the convention that lower rank is better).

The average word likelihood and average rank across all documents in our test set are shown in Table 1. These results are consistent with the values reported in the literature [4, 5]; in most cases CTM performs best, followed by LDA.

### 4.3 Analyzing human evaluations

The tasks described in Section 3 were offered on Amazon Mechanical Turk (http://www.mturk.com), which allows workers (our pool of prospective subjects) to perform small jobs for a fee through a Web interface. No specialized training or knowledge is typically expected of the workers. Amazon Mechanical Turk has been successfully used in the past to develop gold-standard data for natural language processing [22] and to label images [23]. For both the word intrusion and topic intrusion

Table 1: Two predictive metrics: predictive log likelihood/predictive rank. Consistent with values reported in the literature, CTM generally performs the best, followed by LDA, then pLSI. The bold numbers indicate the best performance in each row.

| CORPUS | TOPICS | LDA | CTM | PLSI |
|---|---|---|---|---|
| NEW YORK TIMES | 50 | **-7.3214 / 784.38** | -7.3335 / 788.58 | -7.3384 / 796.43 |
| | 100 | -7.2761 / 778.24 | **-7.2647 / 762.16** | -7.2834 / 785.05 |
| | 150 | -7.2477 / 777.32 | -7.2467 / **755.55** | **-7.2382** / 770.36 |
| WIKIPEDIA | 50 | **-7.5257** / 961.86 | -7.5332 / **936.58** | -7.5378 / 975.88 |
| | 100 | -7.4629 / 935.53 | **-7.4385 / 880.30** | -7.4748 / 951.78 |
| | 150 | -7.4266 / 929.76 | **-7.3872 / 852.46** | -7.4355 / 945.29 |

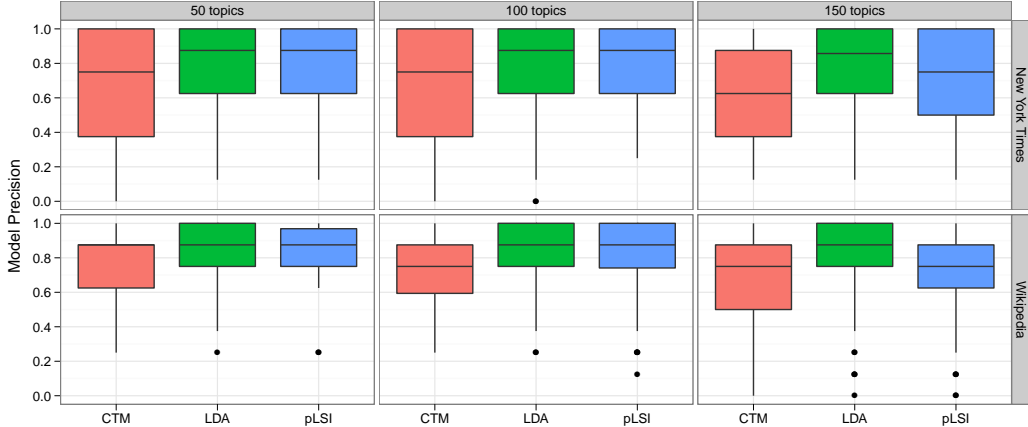

Figure 3: The model precision (Equation 1) for the three models on two corpora. Higher is better. Surprisingly, although CTM generally achieves a better predictive likelihood than the other models (Table 1), the topics it infers fare worst when evaluated against human judgments.

tasks, we presented each worker with jobs containing ten of the tasks described in Section 3. Each job was performed by 8 separate workers, and workers were paid between $0.07 – $0.15 per job.

**Word intrusion**  As described in Section 3.1, the word intrusion task measures how well the inferred topics match human concepts (using *model precision*, i.e., how well the intruders detected by the subjects correspond to those injected into ones found by the topic model).

Let $\omega_k^m$ be the index of the intruding word among the words generated from the $k^{th}$ topic inferred by model $m$. Further let $i_{k,s}^m$ be the intruder selected by subject $s$ on the set of words generated from the $k$th topic inferred by model $m$ and let $S$ denote the number of subjects. We define model precision by the fraction of subjects agreeing with the model,

$$\mathrm{MP}_k^m = \sum_s \mathbb{1}(i_{k,s}^m = \omega_k^m)/S. \tag{1}$$

Figure 3 shows boxplots of the precision for the three models on the two corpora. In most cases LDA performs best. Although CTM gives better predictive results on held-out likelihood, it does not perform as well on human evaluations. This may be because CTM finds correlations between topics and correlations within topics are confounding factors; the intruder for one topic might be selected from another highly correlated topic. The performance of pLSI degrades with larger numbers of topics, suggesting that overfitting [4] might affect interpretability as well as predictive power.

Figure 4 (left) shows examples of topics with high and low model precisions from the NY Times data fit with LDA using 50 topics. In the example with high precision, the topic words all coherently express a painting theme. For the low precision example, "taxis" did not fit in with the other political words in the topic, as $87.5\%$ of subjects chose "taxis" as the intruder.

The relationship between model precision, $\mathrm{MP}_k^m$, and the model's estimate of the likelihood of the intruding word in Figure 5 (top row) is surprising. The highest probability did not have the best interpretability; in fact, the trend was the opposite. This suggests that as topics become more fine-grained in models with larger number of topics, they are less useful for humans. The downward

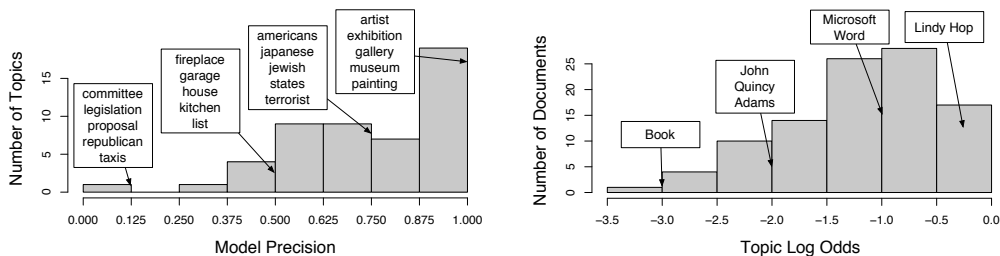

Figure 4: A histogram of the model precisions on the New York Times corpus (left) and topic log odds on the Wikipedia corpus (right) evaluated for the fifty topic LDA model. On the left, example topics are shown for several bins; the topics in bins with higher model precision evince a more coherent theme. On the right, example document titles are shown for several bins; documents with higher topic log odds can be more easily decomposed as a mixture of topics.

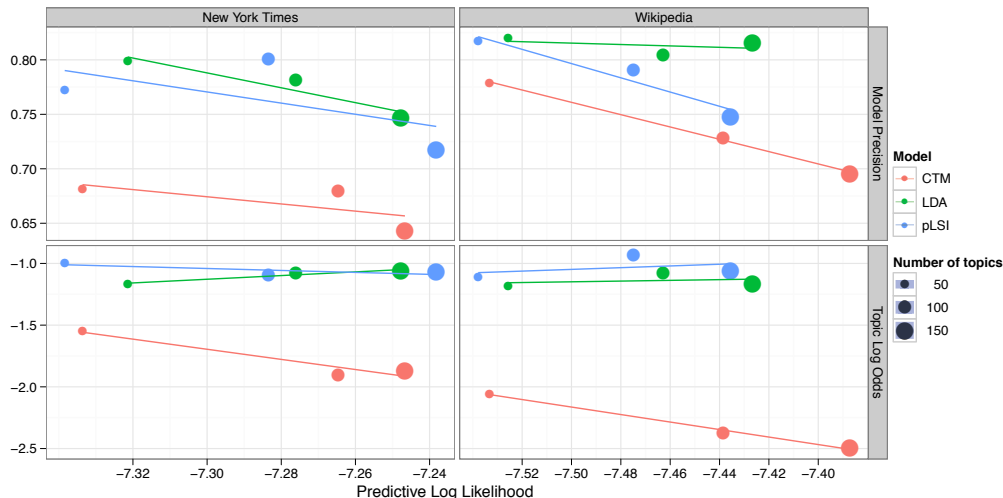

Figure 5: A scatter plot of model precision (top row) and topic log odds (bottom row) vs. predictive log likelihood. Each point is colored by model and sized according to the number of topics used to fit the model. Each model is accompanied by a regression line. Increasing likelihood does not increase the agreement between human subjects and the model for either task (as shown by the downward-sloping regression lines).

sloping trend lines in Figure 5 implying that the models are often trading improved likelihood for lower interpretability.

The model precision showed a negative correlation (Spearman's $\rho = -0.235$ averaged across all models, corpora, and topics) with the number of senses in WordNet of the words displayed to the subjects [24] and a slight positive correlation ($\rho = 0.109$) with the average pairwise Jiang-Conrath similarity of words[1] [25].

**Topic intrusion** In Section 3.2, we introduced the topic intrusion task to measure how well a topic model assigns topics to documents. We define the *topic log odds* as a quantitative measure of the agreement between the model and human judgments on this task. Let $\theta_d^m$ denote model $m$'s point estimate of the topic proportions vector associated with document $d$ (as described in Section 4.2). Further, let $j_{d,s}^m \in \{1 \ldots K\}$ be the intruding topic selected by subject $s$ for document $d$ on model $m$ and let $j_d^m$ denote the "true" intruder, i.e., the one generated by the model. We define the topic log odds as the log ratio of the probability mass assigned to the true intruder to the probability mass

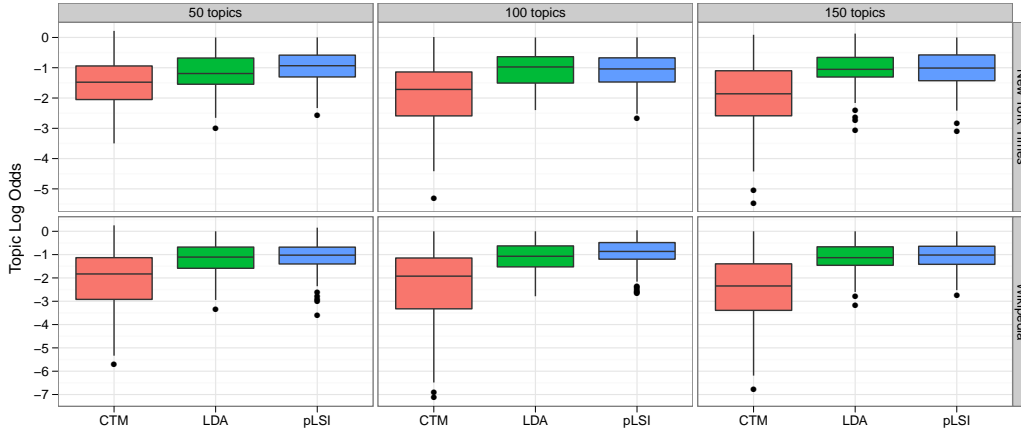

Figure 6: The topic log odds (Equation 2) for the three models on two corpora. Higher is better. Although CTM generally achieves a better predictive likelihood than the other models (Table 1), the topics it infers fare worst when evaluated against human judgments.

assigned to the intruder selected by the subject,

$$\text{TLO}_d^m = (\textstyle\sum_s \log \hat{\theta}_{d,j_{d,*}^m}^m - \log \hat{\theta}_{d,j_{d,s}^m}^m)/S. \tag{2}$$

The higher the value of $\text{TLO}_d^m$, the greater the correspondence between the judgments of the model and the subjects. The upper bound on $\text{TLO}_d^m$ is 0. This is achieved when the subjects choose intruders with a mixture proportion no higher than the true intruder's.

Figure 6 shows boxplots of the topic log odds for the three models. As with model precision, LDA and pLSI generally outperform CTM. Again, this trend runs counter to CTM's superior performance on predictive likelihood. A histogram of the TLO of individual Wikipedia documents is given in Figure 4 (right) for the fifty-topic LDA model. Documents about very specific, unambiguous concepts, such as "Lindy Hop," have high TLO because it is easy for both humans and the model to assign the document to a particular topic. When documents express multiple disparate topics, human judgments diverge from those of the model. At the low end of the scale is the article "Book" which touches on diverse areas such as history, science, and commerce. It is difficult for LDA to pin down specific themes in this article which match human perceptions.

Figure 5 (bottom row) shows that, as with model precision, increasing predictive likelihood does not imply improved topic log odds scores. While the topic log odds are nearly constant across all numbers of topics for LDA and pLSI, for CTM topic log odds and predictive likelihood are negatively correlated, yielding the surprising conclusion that higher predictive likelihoods do not lead to improved model interpretability.

## 5   Discussion

We presented the first validation of the assumed coherence and relevance of topic models using human experiments. For three topic models, we demonstrated that traditional metrics do not capture whether topics are coherent or not. Traditional metrics are, indeed, negatively correlated with the measures of topic quality developed in this paper. Our measures enable new forms of model selection and suggest that practitioners developing topic models should thus focus on evaluations that depend on real-world task performance rather than optimizing likelihood-based measures.

In a more qualitative vein, this work validates the use of topics for corpus exploration and information retrieval. Humans appreciate the semantic coherence of topics and can associate the same documents with a topic that a topic model does. An intriguing possibility is the development of models that explicitly seek to optimize the measures we develop here either by incorporating human judgments into the model-learning framework or creating a computational proxy that simulates human judgments.

**Acknowledgements**

David M. Blei is supported by ONR 175-6343, NSF CAREER 0745520 and grants from Google and Microsoft. We would also like to thank Dan Osherson for his helpful comments.

## Footnotes

[1]Words without entries in WordNet were ignored; polysemy was handled by taking the maximum over all senses of words. To handle words in the same synset (e.g. "fight" and "battle"), the similarity function was capped at 10.0.

# References

[1] Blei, D., J. Lafferty. *Text Mining: Theory and Applications*, chap. Topic Models. Taylor and Francis, 2009.

[2] Mimno, D., A. Mccallum. Organizing the OCA: learning faceted subjects from a library of digital books. In *JCDL*. 2007.

[3] Hofmann, T. Probabilistic latent semantic analysis. In *UAI*. 1999.

[4] Blei, D., A. Ng, M. Jordan. Latent Dirichlet allocation. *JMLR*, 3:993–1022, 2003.

[5] Blei, D. M., J. D. Lafferty. Correlated topic models. In *NIPS*. 2005.

[6] Landauer, T., S. Dumais. Solutions to Plato's problem: The latent semantic analsyis theory of acquisition, induction, and representation of knowledge. *Psychological Review*, 2(104):211–240, 1997.

[7] Landauer, T. K. On the computational basis of learning and cognition: Arguments from LSA. *The Psychology of Learning and Motivation*, 41:43–84, 2002.

[8] Deerwester, S., S. Dumais, T. Landauer, et al. Indexing by latent semantic analysis. *Journal of the American Society of Information Science*, 41(6):391–407, 1990.

[9] Berry, M. W., S. T. Dumais, T. A. Letsche. Computational methods for intelligent information access. In *Supercomputing*. 1995.

[10] Titov, I., R. McDonald. A joint model of text and aspect ratings for sentiment summarization. In *HLT*. 2008.

[11] Wei, X., B. Croft. LDA-based document models for ad-hoc retrieval. In *SIGIR*. 2006.

[12] Boyd-Graber, J. L., D. M. Blei, X. Zhu. Probabalistic walks in semantic hierarchies as a topic model for WSD. In *HLT*. 2007.

[13] Wallach, H. M., I. Murray, R. Salakhutdinov, et al. Evaluation methods for topic models. In *ICML*. 2009.

[14] Griffiths, T., M. Steyvers. Probabilistic topic models. In T. Landauer, D. McNamara, S. Dennis, W. Kintsch, eds., *Latent Semantic Analysis: A Road to Meaning*. Laurence Erlbaum, 2006.

[15] Hall, D., D. Jurafsky, C. D. Manning. Studying the history of ideas using topic models. In *EMNLP*. 2008.

[16] Mei, Q., X. Shen, C. Zhai. Automatic labeling of multinomial topic models. In *KDD*. 2007.

[17] Erosheva, E., S. Fienberg, J. Lafferty. Mixed-membership models of scientific publications. *PNAS*, 101(Suppl 1):5220 — 5227, 2004.

[18] Dempster, A., N. Laird, D. Rubin, et al. Maximum likelihood from incomplete data via the EM algorithm. *Journal of the Royal Statistical Society. Series B*, 39(1):1–38, 1977.

[19] Schmid, H. Probabilistic part-of-speech tagging using decision trees. In *Proceedings of International Conference on New Methods in Language Processing*. 1994.

[20] Loper, E., S. Bird. NLTK: the natural language toolkit. In *Proceedings of the ACL-02 Workshop on Effective tools and methodologies for teaching natural language processing and computational linguistics*. 2002.

[21] Teh, Y. W., K. Kurihara, M. Welling. Collapsed variational inference for HDP. In *NIPS*. 2008.

[22] Snow, R., B. O'Connor, D. Jurafsky, et al. Cheap and fast—but is it good? evaluating non-expert annotations for natural language tasks. In *EMNLP*. 2008.

[23] Deng, J., W. Dong, R. Socher, et al. ImageNet: A Large-Scale Hierarchical Image Database. In *CVPR*. 2009.

[24] Miller, G. A. Nouns in WordNet: A lexical inheritance system. *International Journal of Lexicography*, 3(4):245–264, 1990.

[25] Jiang, J. J., D. W. Conrath. Semantic similarity based on corpus statistics and lexical taxonomy. In *Proceedings on International Conference on Research in Computational Linguistics*. 1997.

